# Statistical and Dynamical Interpretation of ISIH Data from Periodically Stimulated Sensory Neurons

**John K. Douglass and Frank Moss**
Department of Biology and Department of Physics
University of Missouri at St. Louis
St. Louis, MO 63121

**Andre Longtin**
Department of Physics
University of Ottawa
Ottawa, Ontario, Canada K1N 6N5

## Abstract

We interpret the time interval data obtained from periodically stimulated sensory neurons in terms of two simple dynamical systems driven by noise with an embedded weak periodic function called the signal: 1) a bistable system defined by two potential wells separated by a barrier, and 2) a FitzHugh-Nagumo system. The implementation is by analog simulation: electronic circuits which mimic the dynamics. For a given signal frequency, our simulators have only two adjustable parameters, the signal and noise intensities. We show that experimental data obtained from the periodically stimulated mechanoreceptor in the crayfish tailfan can be accurately approximated by these simulations. Finally, we discuss *stochastic resonance* in the two models.

## 1 INTRODUCTION

It is well known that sensory information is transmitted to the brain using a code which must be based on the time intervals between neural firing events or the mean firing rate. However, in any collection of such data, and even when the sensory system is stimulated with a periodic signal, statistical analyses have shown that a significant fraction of the intervals are random, having no coherent relationship to the stimulus. We call this component the "noise". It is clear that coherent and incoherent subsets of such data must be separated. Moreover, the noise intensity depends upon the stimulus intensity in a nonlinear manner through, for example, efferent connections in the visual system (Kaplan and Barlow, 1980) and is often much larger (sometimes several orders of magnitude larger!) than can be accounted for by equilibrium statistical mechanics (Denk and Webb, 1992). Evidence that the noise in networks of neurons can dynamically alter the properties of the membrane potential and time constants has also been accumulated (Kaplan and Barlow, 1976; Treutlein and Schulten, 1985; Bernander, Koch and Douglas, 1992). Recently, based on comparisons of interspike interval histograms (ISIH's) obtained from passive analog simulations of simple bistable systems, with those from auditory neurons, it was suggested that the noise intensity may play a critical role in the ability of the living system to sense the stimulus intensity (Longtin, Bulsara and Moss, 1991). In this work, it is shown that in the simulations, ISIH's are reproduced provided that noise is added to a weak signal, i.e. one that cannot cause firing by itself. All of these processes are essentially nonlinear, and they indicate the ultimate futility of simply measuring the "background spontaneous rate" and later subtracting it from spike rates obtained with a stimulus applied. Indeed, they raise serious doubts regarding the applicability of any *linear* transform theory to neural problems.

In this paper, we investigate the possibility that the noise can enhance the ability of a sensory neuron to transmit information about periodic stimuli. The present study relies on two objects, the ISIH and the power spectrum, both familiar measurements in electrophysiology. These are obtained from analog simulations of two simple dynamical systems, 1) the overdamped motion of a particle in a bistable, quartic potential; and 2) the FitzHugh-Nagumo model. The results of these simulations are compared with those from experiments on the mechanoreceptor in the tailfan of the crayfish *Procambarus clarkii*.

## 2  THE ANALOG SIMULATOR

Previously, we made detailed comparisons of ISIH's obtained from a variety of sensory modalities (Longtin, Bulsara and Moss, 1991) with those measured on the bistable system,

$$\dot{x} = x - x^3 + \xi(t) + \epsilon \sin(\omega t) \tag{1}$$

where $\epsilon$ is the stimulus intensity, and $\xi$ is a quasi white, Gaussian noise, defined by $\langle \xi(t)\xi(s) \rangle = (D/\tau)\exp(- |t\text{-}s|/\tau)$ with $D$ the noise intensity and $\tau$ a (dimensionless) noise correlation time. Quasi white means that the actual noise correlation time is at least one order of magnitude smaller than the integrator

time constant (the "clock" by which the simulator measures time). It was shown that the neurophysiological data could be satisfactorily matched by data from the simulation by adjusting *either* the noise intensity *or* the stimulus intensity provided that the other quantity had a value not very different from the height of the potential barrier. Moreover, bistable dynamical systems of the type represented by Eq. (1) (and many others as well) have been frequently used to demonstrate *stochastic resonance* (SR), an essentially **nonlinear** process whereby the signal-to-noise ratio (SNR) of a weak signal can be enhanced by the noise. Below we show that SR can be demonstrated in a typical excitable system of the type often used to model sensory neurons. This raises a tantalizing question: *can SR be discovered as a naturally occurring phenomenon in living systems?* More information can be found in a recent review and workshop proceedings (Moss, 1993; Chialvo and Apkarian, 1993; Longtin, 1993).

There is, however, a significant difference between the dynamics represented by Eq. (1) and the more usual neuron models which are excitable systems. A simple example of the latter is the FitzHugh-Nagumo (FN) model, the ISIH's of which have recently been studied (Longtin, 1993). The FN model is an excitable system controlled by a bifurcation parameter. When the voltage variable is perturbed past a certain boundary, a large excursion, identified with a neural firing event, occurs. Thus a *deterministic* refractory period is built into the model as the time required for the execution of a single firing event. By contrast, in the bistable system, a firing event is represented by the transition from well $A$ to well $B$. Before another firing can occur, the system must be reset by a reverse transition from $B$ to $A$, which is essentially stochastic. The bistable system thus exhibits *a statistical distribution* of refractory periods. The FN system is not bistable, but, depending on the value of the bifurcation parameter, it can be either periodically firing (oscillating) or residing on a fixed point. The FN model used here is defined by (Longtin, 1993),

$$\dot{v} = v(v - 0.5)(1 - v) - w + \xi(t), \tag{2}$$

$$\dot{w} = v - w - [b + \epsilon \sin(\omega t)], \tag{3}$$

where $v$ is the fast variable (action potential) to which the noise $\xi$ is added, $w$ is the recovery variable to which the signal is added, and $b$ is the bifurcation parameter. The range of behaviors is given by: $b > 0.65$, fixed point and $b \geq 0.65$, oscillating. We operate far into the fixed point regime at $b = 0.9$, so that bursts of sustained oscillations do not occur. Thus single spikes at more-or-less random times but with some coherence with the signal are generated. A schematic diagram of the analog simulator is shown in Fig. 1. The simulator is constructed of standard electronic chips: voltage multipliers ($X$) and operational

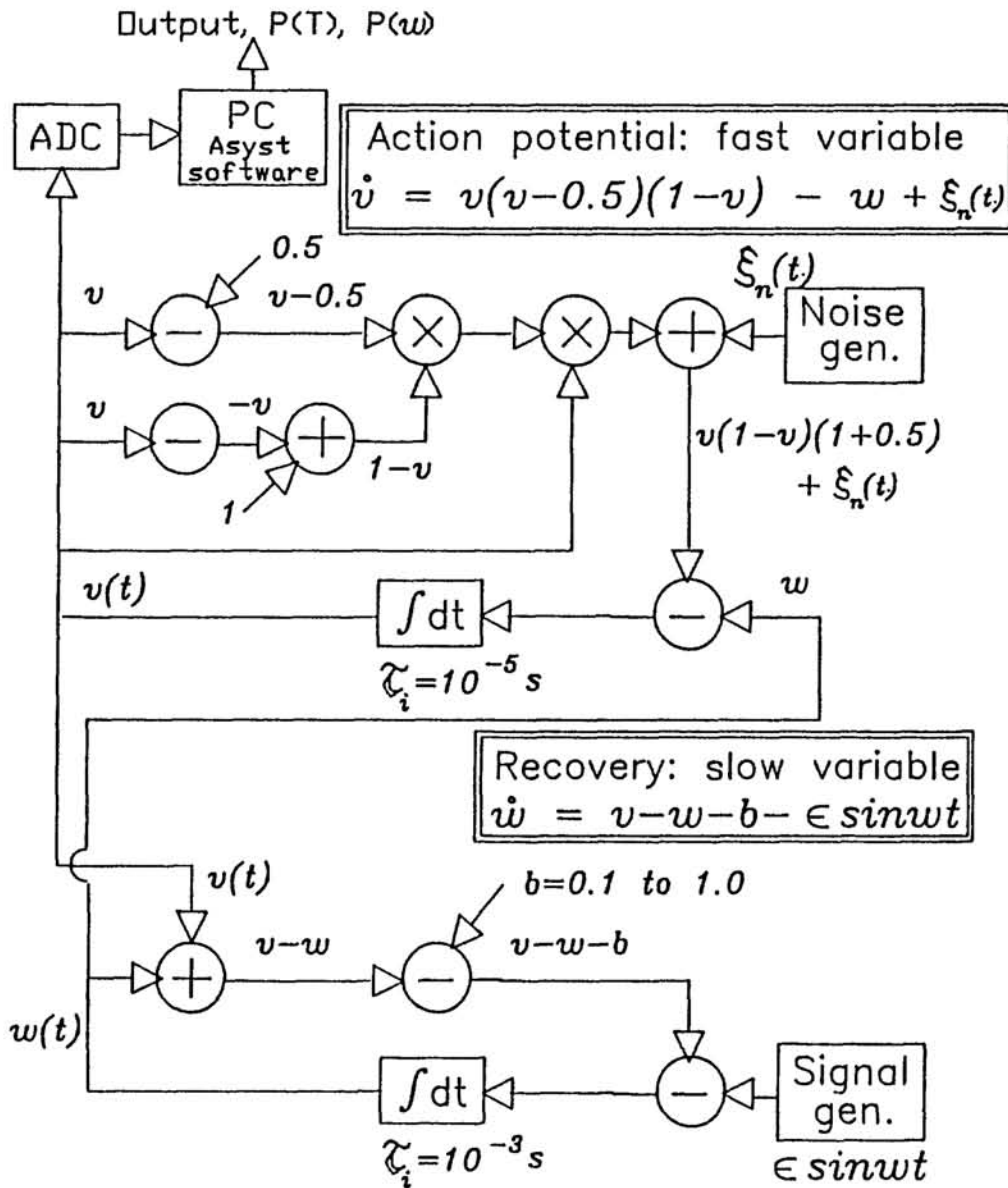

Fig.1 Analog simulator of FitzHugh-Nagumo model. The characteristic response times are determined by the integrator time constants as shown. The noise correlation time was $\tau_n = 10^{-5}$ s.

amplifier summers (±). The fast variable, $v(t)$, was digitized and analyzed for the ISIH and the power spectrum by the PC shown. Note that the noise correlation time, $10^{-5}$ s, is equal to the fast variable integrator time constant and is much larger than the slow variable time constant. This noise is, therefore, colored. Analog simulator designs, nonlinear experiments and colored noise have recently been reviewed (Moss and McClintock, 1989). Below we compare data from this simulator with electrophysiological data from the crayfish.

## 3  EXPERIMENTS WITH CRAYFISH MECHANORECEPTOR CELLS

Single hair mechanoreceptor cells of the crayfish tailfan represent a simple and robust system lacking known efferents. A simple system is necessary, since we are searching for a specific dynamical behavior which might be masked in a more complex physiology. In this system, small motions of the hairs (as small as a few tens of nanometers) are transduced into spike trains which travel up the sensory neuron to the caudal ganglion. These neurons show a range of spontaneous firing rates (internal noise). In this experiment, a neuron with a relatively high internal noise was chosen. Other experiments and more details are described elsewhere (Bulsara, Douglass and Moss, 1993). The preparation consisted of a piece of the tailfan from which the sensory nerve bundle and ganglion were exposed surgically. This appendage was sinusoidally moved through the saline solution by an electromagnetic transducer. Extracellular recordings from an identified hair cell were made using standard methods. The preparations typically persisted in good physiological condition for 8 to 12 hours. An example ISIH is shown in the upper panel of Fig. 2. The stimulus period was, $T_0 = 14\,ms$. Note the peak sequence at the integer multiples of $T_0$ (Longtin, *et al*, 1991). This ISIH was measured in about 15 minutes for which about $8K$ spikes were obtained. An ISIH obtained from the FN simulator in the same time and including about the same number of spikes is shown in the lower panel. The similarity demonstrates that neurophysiological ISIH's can easily be mimicked with FN models as well as with bistable models. Our model is also able to reproduce non renewal effects (data not shown) which occur at high frequency and/or low stimulus or noise intensity, and for which the first peak in the ISIH is *not* the one of maximum amplitude.

We turn now to the question of whether SR, based on the power spectrum, can be demonstrated in such excitable systems. The power spectrum typically shows a sharp peak due to the signal at frequency $\omega_0$, riding on a broad noise background. An example, measured on the FN simulator, is shown in the left panel in Fig. 3. This spectrum was obtained for a constant signal intensity set just above threshold and for the stated external noise intensity. The SNR, in decibels, is defined as the ratio of the strength $S(\omega)$ of the signal feature to the noise amplitude, $N(\omega)$, measured at the base of the signal feature: SNR = 10 $\log_{10}S/N$. The panel on the right of Fig. 3 shows the SNR's obtained from a large number of such power spectra, each measured for a different noise intensity. Clearly there is an optimal noise intensity which maximizes the SNR. This is, to our knowledge, the first demonstration of SR based on the power spectra in an excitable system. Just as for the bistable systems (Moss, 1993), when the external noise intensity is too low, the signal is not "sampled" frequently enough and the SNR is low. By contrast, when the noise intensity is too

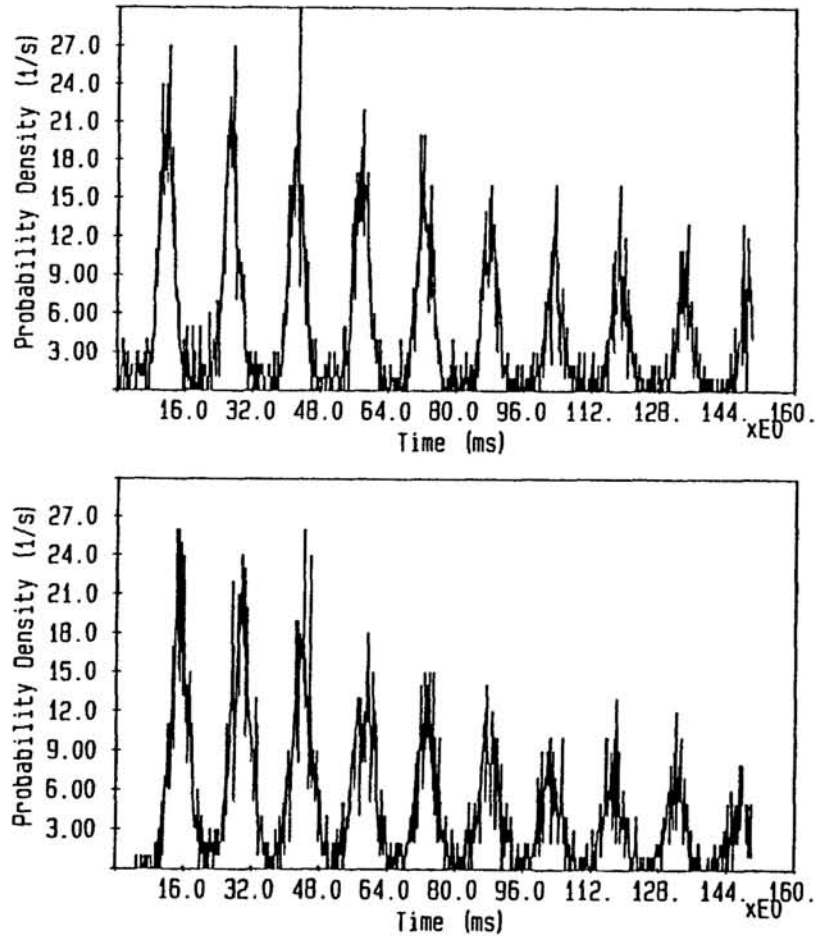

Fig. 2. ISIH's obtained from the crayfish stimulated at 68.6 Hz (upper); and the FN simulator driven at the same frequency with $b = 0.9$, $V_{noise} = 0.022$ $V_{rms}$, and $V_{sig} = 0.53$ $V_{rms}$ (lower).

high, the signal becomes randomized. The occurrence of a maximum in the SNR is thus motivated. SR has also been studied using well residence time probability densities, which are analogous to the physiological ISIH's (Longtin, *el al*, 1991), and was further studied in the FN system (Longtin, 1993). In these cases, it is observed that the individual peak heights pass through maxima as the noise intensity is varied, thus demonstrating SR, similar to that shown in Fig. 3, based on the ISIH (or residence time probability density).

## 4  DISCUSSION

We have shown that physiological measurements such as the familiar ISIH patterns obtained from periodically stimulated sensory neurons can be easily mimicked by analog simulations of simple noisy systems, in particular bistable sys-

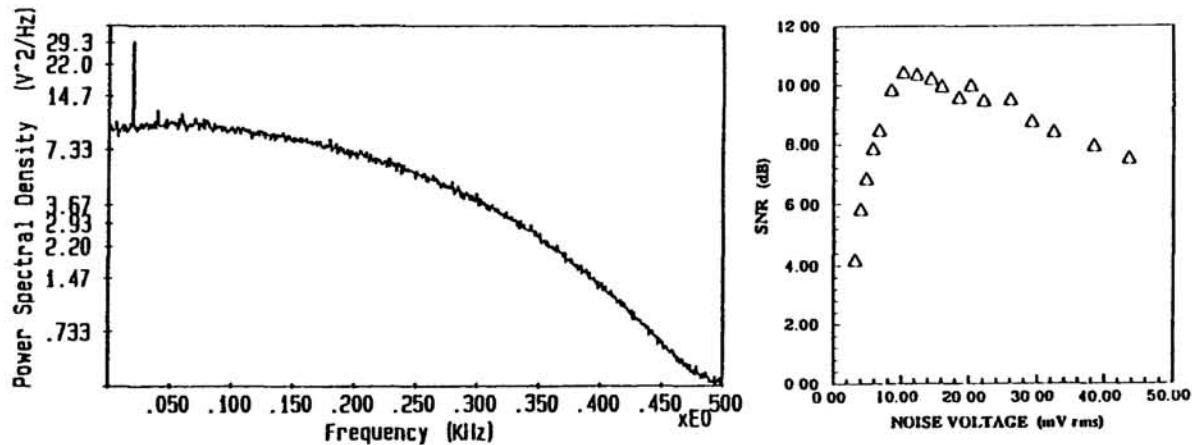

Fig. 3. A power spectrum from the FN simulator stimulated by a 20 Hz signal for $b = 0.9$, $\epsilon = 0.25$ $V$ and $V_{noise} = 0.021$ $V_{rms}$ (left). The SNR's versus noise voltage measured in the FN system showing SR at $V_{noise} \simeq 10$ $mV_{rms}$ (right). Similar SR results based on the ISIH have been obtained by Longtin (1993) and by Chialvo and Apkarian (1993).

tems for which the refractory period is strictly stochastic and excitable systems for which the refractory period is deterministic. Further, we have shown that SR, based on SNR's obtained from the power spectrum, can be demonstrated for the FitzHugh-Nagumo model. It is worth emphasizing that these results are possible only because the systems are inherently nonlinear. The signal alone is too weak to cause firing events in either the bistable or the excitable models. Thus these results suggest that biological systems may be able to detect weak stimuli in the presence of background noise which they could not otherwise detect. Careful behavioral studies will be necessary to decide this question, however, a recent and interesting psychophysics experiment using human interpretations of ambiguous figures, presented in sequences with both coherent and random components points directly to this possibility (Chialvo and Apkarian, 1993).

## Acknowledgements

This work was supported by the Office of Naval Research grant N00014-92-J-1235 and by NSERC (Canada).

## References

Bernander, Ö, Koch, C. and Douglas R. (1992) Network activity determines spatio-temporal integration in single cells, in *Advances in Neural Information Processing Systems 3*; R. Lippman, J. Moody and D. Touretzky, editors; Morgan Kaufmann, San Mateo, CA. 43-50

Bulsara, A., Douglass, J. and Moss, F. (1993) Nonlinear Resonance: Noise-assisted information processing in physical and neurophysiological systems. Nav. Res. Rev. in press.

Chialvo, D. and Apkarian, V. (1993) Modulated noisy biological dynamics: three examples;  in *Proceedings of the NATO ARW on Stochastic Resonance in Physics and Biology*, edited by  F. Moss, A. Bulsara, and M. F. Shlesinger, J. Stat. Phys. **70**, forthcoming

Denk, W. and Webb, W. (1992) Forward and reverse transduction at the limit of sensitivity studied by correlating electrical and mechanical fluctuations in frog saccular hair cells. Hear. Res. **60**, 89-102.

Kaplan, E. and Barlow, R. (1976) Energy, quanta and *Limulus* vision. Vision Res. **16**, 745-751

Kaplan, E. and Barlow, R. (1980) Circadian clock in *Limulus* brain increases response and decreases noise of retinal photoreceptors. Nature **286**, 393

Longtin, A. (1993) Stochastic resonance in neuron models, in *Proceedings of the NATO ARW on Stochastic Resonance in Physics and Biology*, edited by  F. Moss, A. Bulsara, and M. F. Shlesinger, J. Stat. Phys. **70**, forthcoming

Longtin, *A*, Bulsara, *A* and Moss F. (1991) Time interval sequences in bistable systems and the noise-induced transmission of information by sensory neurons. Phys. Rev. Lett. **67**, 656-659

Moss, F. (1993) Stochastic resonance: from the ice ages to the monkey's ear; in, *Some Problems in Statistical Physics*, edited by G. H. Weiss, SIAM, Philadelphia, in press

Moss, F. and McClintock, P.V.E. editors (1989) *Noise in Nonlinear Dynamical Systems, Vols. 1 - 3*, Cambridge University Press.

Treutlein, H. and Schulten, K. (1985) Noise induced limit cycles of the Bonhoeffer-Van der Pol model of neural pulses. Ber. Bunsenges. Phys. Chem. **89**, 710.
